# HIGH ORDER NEURAL NETWORKS FOR EFFICIENT
# ASSOCIATIVE MEMORY DESIGN

I. GUYON*, L. PERSONNAZ*, J. P. NADAL** and G. DREYFUS*

\* Ecole Supérieure de Physique et de Chimie Industrielles de la Ville de Paris
Laboratoire d'Electronique
10, rue Vauquelin
75005 Paris (France)

\*\* Ecole Normale Supérieure
Groupe de Physique des Solides
24, rue Lhomond
75005 Paris (France)

## ABSTRACT

We propose learning rules for recurrent neural networks with high-order interactions between some or all neurons. The designed networks exhibit the desired associative memory function : perfect storage and retrieval of pieces of information and/or sequences of information of any complexity.

## INTRODUCTION

In the field of information processing, an important class of potential applications of neural networks arises from their ability to perform as associative memories. Since the publication of J. Hopfield's seminal paper[1], investigations of the storage and retrieval properties of recurrent networks have led to a deep understanding of their properties. The basic limitations of these networks are the following :
- their storage capacity is of the order of the number of neurons ;
- they are unable to handle structured problems ;
- they are unable to classify non-linearly separable data.

In order to circumvent these limitations, one has to introduce additional non-linearities. This can be done either by using "hidden", non-linear units, or by considering multi-neuron interactions[2]. This paper presents learning rules for networks with multiple interactions, allowing the storage and retrieval, either of static pieces of information (autoassociative memory), or of temporal sequences (associative memory), while preventing an explosive growth of the number of synaptic coefficients.

## AUTOASSOCIATIVE MEMORY

The problem that will be addressed in this paragraph is how to design an autoassociative memory with a recurrent (or feedback) neural network when the number p of prototypes is large as compared to the number n of neurons. We consider a network of n binary neurons, operating in a synchronous mode, with period $\tau$. The state of neuron i at time t is denoted by $\sigma_i(t)$, and the state of the network at time t is represented by a vector $\underline{\sigma}(t)$ whose components are the $\sigma_i(t)$. The dynamics of each neuron is governed by the following relation :

$$\sigma_i(t+\tau) = \text{sgn } v_i(t). \tag{1}$$

In networks with two-neuron interactions only, the potential $v_i(t)$ is a linear function of the state of the network :

$$v_i(t) = \sum_{j=1}^{n} C_{ij}\, \sigma_j(t).$$

For autoassociative memory design, it has been shown[3] that any set of correlated patterns, up to a number of patterns p equal to $2^n$, can be made the stable states of the system, provided the synaptic matrix is computed as the orthogonal projection matrix onto the subspace spanned by the stored vectors. However, as p increases, the rank of the family of prototype vectors will increase, and finally reach the value of n. In such a case, the synaptic matrix reduces to the identity matrix, so that all $2^n$ states are stable and the energy landscape becomes flat. Even if such an extreme case is avoided, the attractivity of the stored states decreases with increasing p, or, in other terms,

the number of fixed points which are not the stored patterns increases ; this problem can be alleviated to a large extent by making a useful use of these "spurious" fixed points[4]. Another possible solution consists in "gardening" the state space in order to enlarge the basins of attraction of the fixed points[5]. Anyway, no dramatic improvements are provided by all these solutions since the storage capacity is always $O(n)$.

We now show that the introduction of high-order interactions between neurons, increases the storage capacity proportionally to the number of connections per neuron. The dynamical behaviour of neuron i is still governed by (1). We consider two and three-neuron interactions, extension to higher order are straightforward.
The potential $v_i(t)$ is now defined as

$$v_i(t) = \sum_j C_{i,j}\, \sigma_j(t) + \sum_{j,l} C_{i,jl}\, \sigma_j(t)\, \sigma_l(t).$$

It is more convenient, for the derivation of learning rules, to write the potential in the matrix form :

$$\underline{v}(t) = C\, \underline{\chi}(t),$$

where $\underline{\chi}(t)$ is an m dimensional vector whose components are taken among the set of the $(n^2+n)/2$ values : $\sigma_1, \dots, \sigma_n, \sigma_1\sigma_2, \dots, \sigma_j\sigma_l, \dots, \sigma_{n-1}\sigma_n$.

As in the case of the two-neuron interactions model, we want to compute the interaction coefficients so that the prototypes are stable and attractor states. A condition to store a set of states $\underline{\sigma}^k$ (k=1 to p) is that $\underline{v}^k = \underline{\sigma}^k$ for all k. Among the solutions, the most convenient solution is given by the (n,m) matrix

$$C = \Sigma\, \Gamma^I \tag{2}$$

where $\Sigma$ is the (n,p) matrix whose columns are the $\underline{\sigma}^k$ and $\Gamma^I$ is the (p,m) pseudoinverse of the (m,p) matrix $\Gamma$ whose columns are the $\underline{\chi}^k$. This solution satisfies the above requirements, up to a storage capacity which is related to the dimension m of vectors $\underline{\chi}$. Thus, in a network with three-neuron

interactions, the number of patterns that can be stored is $O(n^2)$. Details on these derivations are published in Ref.6.

By using only a subset of the products $\{\sigma_i\,\sigma_j\}$, the increase in the number of synaptic coefficients can remain within acceptable limits, while the attractivity of the stored patterns is enhanced, even though their number exceeds the number of neurons ; this will be examplified in the simulations presented below.

Finally, it can be noticed that, if vector $\chi$ contains all the $\{\sigma_i\,\sigma_j\}$, i=1,...n, j=1,...n, only, the computation of the vector potential $\underline{v}=C\chi$ can be performed after the following expression :

$$\underline{v} \;=\; \Sigma\; \{\,(\Sigma^T\Sigma)^{②}\}^{I}\;\; (\Sigma^T\underline{\sigma})^{②}$$

where ② stands for the operation which consists in squaring all the matrix coefficients. Hence, the computation of the synaptic coefficients is avoided, memory and computing time are saved if the simulations are performed on a conventional computer. This formulation is also meaningful for optical implementations, the function ② being easily performed in optics[7].

In order to illustrate the capabilities of the learning rule, we have performed numerical simulations which show the increase of the size of the basins of attraction when second-order interactions, in addition to the first-order ones, are used. The simulations were carried out as follows. The number of neurons n being fixed, the amount of second-order interactions was chosen ; p prototype patterns were picked randomly, their components being ±1 with probability 0.5 ; the second-order interactions were chosen randomly. The synaptic matrix was computed from relation (2). The neural network was forced into an initial state lying at an initial Hamming distance $H_i$ from one of the prototypes $\underline{\sigma}^k$ ; it was subsequently left to evolve until it reached a stable state at a distance $H_f$ from $\underline{\sigma}^k$. This procedure was repeated many times for each prototype and the $H_f$ were averaged over all the tests and all the prototypes.

Figures 1a. and 1b. are charts of the mean values of $H_f$ as a function of the number of prototypes, for n = 30 and for various values of m (the dimension of

vector $\chi$). These curves allowed us to determine the maximum number of prototype states which can be stored for a given quality of recall. Perfect recall implies $H_f = 0$ ; when the number of prototypes increases, the error in recall may reach $H_f \approx H_i$ : the associative memory is degenerate. The results obtained for $H_i / n = 10\%$ are plotted on Figure 1a. When no high-order interactions were used, $H_f$ reached $H_i$ for $p/n \approx 1$, as expected ; conversely, virtually no error in recall occured up to $p/n \approx 2$ when all second-order interactions were taken into account ($m = 465$). Figure 1b shows the same quantities for $H_i = 20\%$ ; since the initial states were more distant from the prototypes, the errors in recall were more severe.

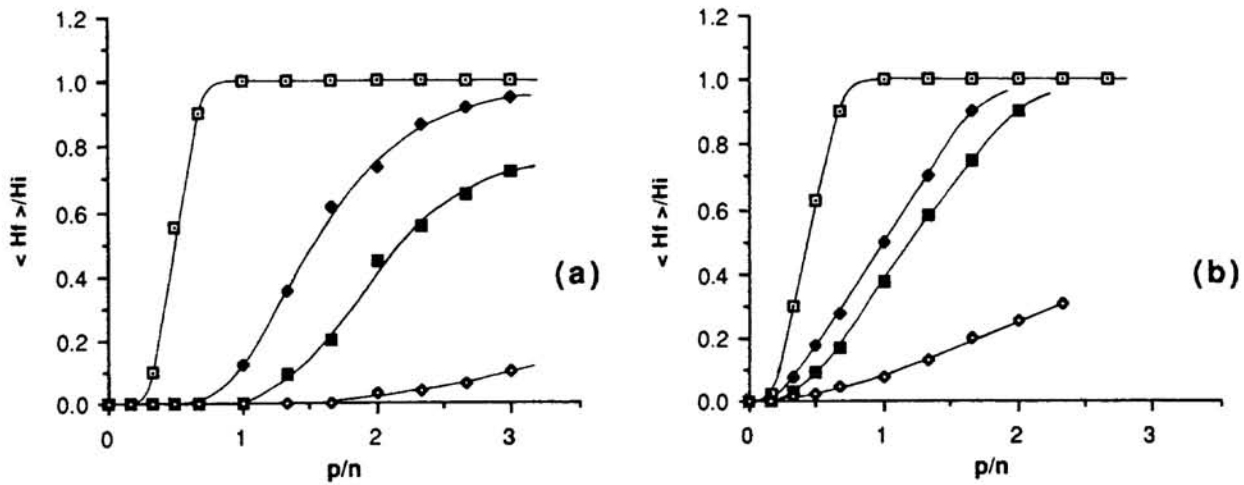

Fig. 1. Improvement of the attractivity by addition of three-neuron interactions to the two-neuron interactions. All prototypes are always stored exactly (all curves go through the origin). Each point corresponds to an average over min(p,10) prototypes and 30 tests for each prototype.
□ Projection : $m = n = 30$ ; ◆ $m = 120$ ; ■ $m = 180$ ; ◇ $m = 465$ (all interactions)
1 a : $H_i / n = 10\%$ ; 1 b : $H_i / n = 20\%$.

## TEMPORAL SEQUENCES (ASSOCIATIVE MEMORY)

The previous section was devoted to the storage and retrieval of items of information considered as fixed points of the dynamics of the network (autoassociative memory design). However, since fully connected neural networks are basically dynamical systems, they are natural candidates for

storing and retrieving information which is dynamical in nature, i.e., temporal sequences of patterns[8]. In this section, we propose a general solution to the problem of storing and retrieving sequences of arbitrary complexity, in recurrent networks with parallel dynamics.

Sequences consist in sets of transitions between states $\underline{\sigma}^k -> \underline{\sigma}^{k+1}$, k=1,..., p. A sufficient condition to store these sets of transitions is that $\underline{v}^k = \underline{\sigma}^{k+1}$ for all k. In the case of a linear potential $\underline{v}=C\ \underline{\sigma}$, the storage prescription  proposed in ref.3 can be used :    $C=\Sigma^+\Sigma^I$ ,

where $\Sigma$ is a matrix whose columns are the  $\underline{\sigma}^k$ and $\Sigma^+$ is the matrix whose columns are the successors  $\underline{\sigma}^{k+1}$ of  $\underline{\sigma}^k$. If p is larger than n, one can use high-order interactions, which leads to introduce a non-linear potential $\underline{v}=C\ \underline{\gamma}$, with $\underline{\gamma}$ as previously defined. We proposed in ref.10 the following storage prescription :

$$C=\Sigma^+\Gamma^I \qquad (3)$$

The two above prescriptions are only valid for storing simple sequences, where no patterns occur twice (or more). Suppose that one pattern occurs twice ; when the network reaches this *bifurcation point,* it is unable to make a decision according the deterministic dynamics described in (1), since the knowledge of the present state is not sufficient. Thus, complex sequences require to keep, at each time step of the dynamics, a non-zero memory span.

The vector potential  $\underline{v}=C\underline{\gamma}$ must involve the states at time t and t-$\tau$, which leads to define the vector $\underline{\gamma}$ as a concatenation of vectors  $\underline{\sigma}(t)$, $\underline{\sigma}(t-\tau)$, $\underline{\sigma}(t)\otimes\underline{\sigma}(t)$, $\underline{\sigma}(t)\otimes\underline{\sigma}(t-\tau)$, or a suitable subset thereof. The subsequent vector $\underline{\sigma}(t+\tau)$ is still determined by  relation (1). In this form, the problem is a generalization of the storage of patterns with high order interactions, as described above. The storage of sequences can be still processed by relation (3).

The solution presented above has the following features :
i)   Sequences with bifurcation points can be stored and retrieved.
ii)  The dimension of the synaptic matrix is at most $(n,2(n^2+n))$, and at least $(n,2n)$ in the linear case, so that at most $2n(n^2+n)$ and at least $2n^2$ synapses are required.

iii) The storage capacity is 0(m), where m is the dimension of the vector $\gamma$.

iv) Retrieval of a sequence requires initializing the network with *two* states in succession.

The example of Figure 2 illustrates the retrieval performances of the latter learning rule. We have limited vector $\gamma$ to $\underline{\sigma}(t) \otimes \underline{\sigma}(t-\tau)$. In a network of n=48 neurons, a large number of poems have been stored, with a total of p=424 elementary transitions. Each state is consists in the 6 bit codes of 8 letters.

|  |  |
|---|---|
| ALOUETTE |  |
| JE TE | **JE NE** |
| PLUMERAI | **OLVMERAI** |
| ALOUETTE | AQFUETTE |
| GENTILLE | JEHKILLE |
| ALOUETTE | SLOUETTE |
| ALOUETTE | ALOUETTE |
| JE TE | JE TE |
| PLUMERAI | PLUMERAI |
| ... | ... |

Fig. 2. One of the stored poems is shown in the first column. The network is initialized with two states (the first two lines of the second column). After a few steps, the network reaches the nearest stored sequence.

## LOCAL LEARNING

Finally, it should be mentioned that all the synaptic matrices introduced in this paper can be computed by iterative, local learning rules.

For autoassociative memory, it has been shown analytically[9] that the procedure :

$$C_{ij}(k) = C_{ij}(k-1) + (1/n) \, ( \sigma_i^k - v_i^k ) \, \sigma_j^k \qquad \text{with } C_{ij}(0) = 0,$$

which is a Widrow-Hoff type learning rule, yields the projection matrix, when

the number of presentations of the prototypes $\{\underline{\sigma}^k\}$ goes to infinity, if the latter are linearly independent.

A derivation along the same lines shows that, by repeated presentations of the prototype transitions, the learning rules :

$$C_{ij}(k) = C_{ij}(k-1) + (1/n) \, ( \, \sigma_i^k - v_i^k \, ) \, \gamma_j^k \qquad \text{with } C_{ij}(0) = 0$$

$$C_{ij}(k) = C_{ij}(k-1) + (1/n) \, ( \, \sigma_i^{k+1} - v_i^k \, ) \, \gamma_j^k \quad \text{with } C_{ij}(0) = 0$$

lead to the exact solutions (relations (2) and (3) respectively), if the vectors $\underline{\gamma}^k$ are linearly independent.

## GENERALIZATION TASKS

Apart from storing and retrieving static pieces of information or sequences, neural networks can be used to solve problems in which there exists a structure or regularity in the sample patterns (for example presence of clumps, parity, symmetry...) that the network must discover. Feed-forward networks with multiple layers of first-order neurons can be trained with back-propagation algorithms for these purposes; however, one-layer feed-forward networks with multi-neuron interactions provide an interesting alternative. For instance, a proper choice of vector $\underline{\gamma}$ (second-order terms only) with the above learning rule yields a perfectly straightforward solution to the exclusive-OR problem. Maxwell et al. have shown that a suitable high-order neuron is able to exhibit the "ad hoc network solution" for the contiguity problem[11].

## CONCLUSION

The use of neural networks with high-order interactions has long been advocated as a natural way to overcome the various limitations of the Hopfield model. However, no procedure guaranteed to store any set of information as fixed points or as temporal sequences had been proposed. The purpose of the present paper is to present briefly such storage prescriptions and show

some illustrations of the use of these methods. Full derivations and extensions will be published in more detailed papers.

## REFERENCES

1. J. J. Hopfield, Proc. Natl. Acad. Sci. (USA) 79, 2554 (1982).
2. P. Peretto and J. J. Niez, Biol. Cybern. 54 , 53 (1986).
   P. Baldi and S. S. Venkatesh, Phys. Rev. Lett. 58 , 913 (1987).
   For more references see ref.6.
3. L. Personnaz, I. Guyon, G. Dreyfus, J. Phys. Lett. 46 , 359 (1985).
   L. Personnaz, I. Guyon, G. Dreyfus, Phys. Rev. A 34 , 4217 (1986).
4. I. Guyon, L. Personnaz, G. Dreyfus, in "Neural Computers", R. Eckmiller and C. von der Malsburg eds (Springer, 1988).
5. E. Gardner, Europhys. Lett. 4, 481 (1987).
   G. Pöppel and U.Krey, Europhys. Lett., 4, 979 (1987).
6. L. Personnaz, I. Guyon, G. Dreyfus, Europhys. Lett. 4, 863 (1987).
7. D. Psaltis and C. H. Park, in "Neural Networks for Computing", J. S. Denker ed., (A.I.P. Conference Proceedings 151, 1986).
8. P. Peretto, J. J. Niez, in "Disordered Systems and Biological Organization", E. Bienenstock, F. Fogelman, G. Weisbush eds (Springer, Berlin 1986).
   S. Dehaene, J. P. Changeux, J. P. Nadal, PNAS (USA) 84, 2727 (1987).
   D. Kleinfeld, H. Sompolinsky, preprint 1987.
   J. Keeler, to appear in J. Cog. Sci.
   For more references see ref. 9.
9. I. Guyon, L. Personnaz, J.P. Nadal and G. Dreyfus, submitted for publication.
10. S. Diederich, M. Opper, Phys. Rev. Lett. 58, 949 (1987).
11. T. Maxwell, C. Lee Giles, Y. C. Lee, Proceedings of ICNN-87, San Diego, 1987.
